# Multiple Alignment of Continuous Time Series

**Jennifer Listgarten**[†]**, Radford M. Neal**[†]**, Sam T. Roweis**[†] **and Andrew Emili**[‡]
[†] Department of Computer Science, [‡] Banting and Best Department of Medical Research
and Program in Proteomics and Bioinformatics
University of Toronto, Toronto, Ontario, M5S 3G4
{jenn,radford,roweis}@cs.toronto.edu, andrew.emili@utoronto.ca

## Abstract

Multiple realizations of continuous-valued time series from a stochastic process often contain systematic variations in rate and amplitude. To leverage the information contained in such noisy replicate sets, we need to align them in an appropriate way (for example, to allow the data to be properly combined by adaptive averaging). We present the Continuous Profile Model (CPM), a generative model in which each observed time series is a non-uniformly subsampled version of a single latent trace, to which local rescaling and additive noise are applied. After unsupervised training, the learned trace represents a canonical, high resolution fusion of all the replicates. As well, an alignment in time and scale of each observation to this trace can be found by inference in the model. We apply CPM to successfully align speech signals from multiple speakers and sets of Liquid Chromatography-Mass Spectrometry proteomic data.

## 1   A Profile Model for Continuous Data

When observing multiple time series generated by a noisy, stochastic process, large systematic sources of variability are often present. For example, within a set of nominally replicate time series, the time axes can be variously shifted, compressed and expanded, in complex, non-linear ways. Additionally, in some circumstances, the scale of the measured data can vary systematically from one replicate to the next, and even within a given replicate.

We propose a Continuous Profile Model (CPM) for simultaneously analyzing a set of such time series. In this model, each time series is generated as a noisy transformation of a single latent trace. The latent trace is an underlying, noiseless representation of the set of replicated, observable time series. Output time series are generated from this model by moving through a sequence of hidden states in a Markovian manner and emitting an observable value at each step, as in an HMM. Each hidden state corresponds to a particular location in the latent trace, and the emitted value from the state depends on the value of the latent trace at that position. To account for changes in the amplitude of the signals across and within replicates, the latent time states are augmented by a set of scale states, which control how the emission signal will be scaled relative to the value of the latent trace. During training, the latent trace is learned, as well as the transition probabilities controlling the Markovian evolution of the scale and time states and the overall noise level of the

observed data. After training, the latent trace learned by the model represents a higher resolution 'fusion' of the experimental replicates. Figure 1 illustrate the model in action.

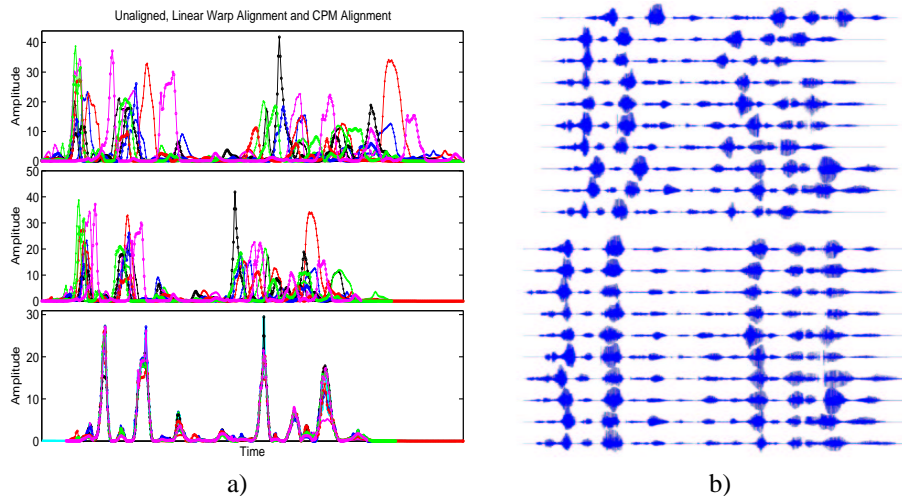

a)                                                                    b)

Figure 1: a) Top: ten replicated speech energy signals as described in Section 4), Middle: same signals, aligned using a linear warp with an offset, Bottom: aligned with CPM (the learned latent trace is also shown in cyan). b) Speech waveforms corresponding to energy signals in a), Top: unaligned originals, Bottom: aligned using CPM.

## 2  Defining the Continuous Profile Model (CPM)

The CPM is generative model for a set of $K$ time series, $\vec{x}^k = (x_1^k, x_2^k, ..., x_{N^k}^k)$. The temporal sampling rate within each $\vec{x}^k$ need not be uniform, nor must it be the same across the different $\vec{x}^k$. Constraints on the variability of the sampling rate are discussed at the end of this section. For notational convenience, we henceforth assume $N^k = N$ for all $k$, but this is not a requirement of the model.

The CPM is set up as follows: We assume that there is a latent trace, $\vec{z} = (z_1, z_2, ..., z_M)$, a canonical representation of the set of noisy input replicate time series. Any given observed time series in the set is modeled as a non-uniformly subsampled version of the latent trace to which local scale transformations have been applied. Ideally, $M$ would be infinite, or at least very large relative to $N$ so that any experimental data could be mapped precisely to the correct underlying trace point. Aside from the computational impracticalities this would pose, great care to avoid overfitting would have to be taken. Thus in practice, we have used $M = (2 + \epsilon)N$ (double the resolution, plus some slack on each end) in our experiments and found this to be sufficient with $\epsilon < 0.2$. Because the resolution of the latent trace is higher than that of the observed time series, experimental time can be made effectively to speed up or slow down by advancing along the latent trace in larger or smaller jumps.

The subsampling and local scaling used during the generation of each observed time series are determined by a sequence of hidden state variables. Let the state sequence for observation $k$ be $\vec{\pi}^k$. Each state in the state sequence maps to a time state/scale state pair: $\pi_i^k \rightarrow \{\tau_i^k, \phi_i^k\}$. Time states belong to the integer set $(1..M)$; scale states belong to an ordered set $(\phi_1..\phi_Q)$. (In our experiments we have used Q=7, evenly spaced scales in logarithmic space). States, $\pi_i^k$, and observation values, $x_i^k$, are related by the emission probability distribution: $A_{\pi_i^k}(x_i^k|\vec{z}) \equiv p(x_i^k|\pi_i^k, \vec{z}, \sigma, u^k) \equiv \mathcal{N}(x_i^k; z_{\tau_i^k}\phi_i^k u^k, \sigma)$, where $\sigma$

is the noise level of the observed data, $\mathcal{N}(a; b, c)$ denotes a Gaussian probability density for $a$ with mean $b$ and standard deviation $c$. The $u^k$ are real-valued scale parameters, one per observed time series, that correct for any overall scale difference between time series $k$ and the latent trace.

To fully specify our model we also need to define the state transition probabilities. We define the transitions between time states and between scale states separately, so that $T^k_{\pi_{i-1}, \pi_i} \equiv p(\pi_i|\pi_{i-1}) = p(\phi_i|\phi_{i-1})p^k(\tau_i|\tau_{i-1})$. The constraint that time must move forward, cannot stand still, and that it can jump ahead no more than $J_\tau$ time states is enforced. (In our experiments we used $J_\tau = 3$.) As well, we only allow scale state transitions between neighbouring scale states so that the local scale cannot jump arbitrarily. These constraints keep the number of legal transitions to a tractable computational size and work well in practice. Each observed time series has its own time transition probability distribution to account for experiment-specific patterns. Both the time and scale transition probability distributions are given by multinomials:

$$
p^k(\tau_i = a|\tau_{i-1} = b) = \begin{cases} d^k_1, & \text{if } a - b = 1 \\ d^k_2, & \text{if } a - b = 2 \\ \vdots \\ d^k_{J_\tau}, & \text{if } a - b = J_\tau \\ 0, & \text{otherwise} \end{cases}
$$

$$
p(\phi_i = a|\phi_{i-1} = b) = \begin{cases} s_0, & \text{if } D(a, b) = 0 \\ s_1, & \text{if } D(a, b) = 1 \\ s_1, & \text{if } D(a, b) = -1 \\ 0, & \text{otherwise} \end{cases}
$$

where $D(a, b) = 1$ means that $a$ is one scale state larger than $b$, and $D(a, b) = -1$ means that $a$ is one scale state smaller than $b$, and $D(a, b) = 0$ means that $a = b$. The distributions are constrained by: $\sum_{i=1}^{J_\tau} d^k_i = 1$ and $2s_1 + s_0 = 1$.

$J_\tau$ determines the maximum allowable instantaneous speedup of one portion of a time series relative to another portion, within the same series or across different series. However, the length of time for which any series can move so rapidly is constrained by the length of the latent trace; thus the maximum overall ratio in speeds achievable by the model between any two entire time series is given by $\min(J_\tau, \frac{M}{N})$.

After training, one may examine either the latent trace or the alignment of each observable time series to the latent trace. Such alignments can be achieved by several methods, including use of the Viterbi algorithm to find the highest likelihood path through the hidden states [1], or sampling from the posterior over hidden state sequences. We found Viterbi alignments to work well in the experiments below; samples from the posterior looked quite similar.

## 3   Training with the Expectation-Maximization (EM) Algorithm

As with HMMs, training with the EM algorithm (often referred to as Baum-Welch in the context of HMMs [1]), is a natural choice. In our model the E-Step is computed exactly using the Forward-Backward algorithm [1], which provides the posterior probability over states for each time point of every observed time series, $\gamma^k_s(i) \equiv p(\pi_i = s|\vec{x})$ and also the pairwise state posteriors, $\xi_{s,t}(i) \equiv p(\pi_{i-1} = s, \pi_i = t|\vec{x}^k)$. The algorithm is modified

only in that the emission probabilities depend on the latent trace as described in Section 2. The M-Step consists of a series of analytical updates to the various parameters as detailed below.

Given the latent trace (and the emission and state transition probabilities), the complete log likelihood of $K$ observed time series, $\vec{x}^k$, is given by $\mathcal{L}^p \equiv \mathcal{L} + \mathcal{P}$. $\mathcal{L}$ is the likelihood term arising in a (conditional) HMM model, and can be obtained from the Forward-Backward algorithm. It is composed of the emission and state transition terms. $\mathcal{P}$ is the log prior (or penalty term), regularizing various aspects of the model parameters as explained below. These two terms are:

$$\mathcal{L} \equiv \sum_{k=1}^{K} \left( \log p(\pi_1) + \sum_{i=1}^{N} \log A_{\pi_i}(x_i^k | \vec{z}) + \sum_{i=2}^{N} \log T_{\pi_{i-1}, \pi_i}^k \right) \tag{1}$$

$$\mathcal{P} \equiv -\lambda \sum_{j=1}^{\tau-1} (z_{j+1} - z_j)^2 + \sum_{k=1}^{K} \log \mathcal{D}(d_v^k | \{\eta_v^k\}) + \log \mathcal{D}(s_v | \{\eta_v'\}), \tag{2}$$

where $p(\pi_1)$ are priors over the initial states. The first term in Equation 2 is a smoothing penalty on the latent trace, with $\lambda$ controlling the amount of smoothing. $\eta_v^k$ and $\eta_v'$ are Dirichlet hyperprior parameters for the time and scale state transition probability distributions respectively. These ensure that all non-zero transition probabilities remain non-zero. For the time state transitions, $v \in \{1, J_\tau\}$ and $\eta_v^k$ corresponds to the pseudo-count data for the parameters $d_1, d_2 \ldots d_{J_\tau}$. For the scale state transitions, $v \in \{0, 1\}$ and $\eta_v^k$ corresponds to the pseudo-count data for the parameters $s_0$ and $s_1$.

Letting $S$ be the total number of possible states, that is, the number of elements in the cross-product of possible time states and possible scale states, the expected complete log likelihood is:

$$<\mathcal{L}^p>_\pi = \mathcal{P} + \sum_{k=1}^{K} \sum_{s=1}^{S} \gamma_s^k(1) \log T_{0,s}^k + \sum_{k=1}^{K} \sum_{s=1}^{S} \sum_{i=1}^{N} \gamma_s^k(i) \log A_s(x_i^k | \vec{z}) + \ldots$$

$$\ldots + \sum_{k=1}^{K} \sum_{s=1}^{S} \sum_{s'=1}^{S} \sum_{i=2}^{N} \xi_{s,s'}^k(i) \log T_{s,s'}^k$$

using the notation $T_{0,s}^k \equiv p(\pi_1 = s)$, and where $\gamma_s^k(i)$ and $\xi_{s,s'}^k(i)$ are the posteriors over states as defined above. Taking derivatives of this quantity with respect to each of the parameters and finding the critical points provides us with the M-Step update equations. In updating the latent trace $\vec{z}$ we obtain a system of $M$ simultaneous equations, for $j = 1..M$:

$$\frac{\partial <\mathcal{L}^p>_\pi}{\partial z_j} = 0 = \sum_{k=1}^{K} \sum_{\{s | \tau_s = j\}} \sum_{i=1}^{N} \left[ \gamma_s^k(i) \phi_s u^k \frac{(x_i^k - z_j u^k \phi_s)}{\sigma^2} \right] - \lambda(4z_j - 2z_{j-1} - 2z_{j+1})$$

For the cases $j = 1, N$, the terms $2z_{j-1}$ and $2z_{j+1}$, respectively, drop out. Considering all such equations we obtain a system of $M$ equations in $M$ unknowns. Each equation depends only linearly on three variables from the latent trace. Thus the solution is easily obtained numerically by solving a tridiagonal linear system.

Analytic updates for $\sigma^2$ and $u^k$ are given by:

$$\sigma^2 = \frac{\sum_{s=1}^{S} \sum_{i=1}^{N} \gamma_s^k(i)(x_i^k - z_{\tau_s} u^k \phi_s)^2}{N}, \quad u^k = \frac{\sum_{s=1}^{S} z_{\tau_s} \phi_s \sum_{i=1}^{N} \gamma_s^k(i) x_i^k}{\sum_{s=1}^{S} (z_{\tau_s} \phi_s)^2 \sum_{i=1}^{N} \gamma_s^k(i)}$$

Lastly, updates for the scale and state transition probability distributions are given by:

$$d_v^k = \frac{\eta_v^k + \sum_{s=1}^{S} \sum_{\{s'|\tau_{s'}-\tau_s=v\}} \sum_{i=2}^{N} \xi_{s,s''}^k(i)}{\sum_{j=1}^{J_\tau} \eta_j^k + \sum_{j=1}^{J_\tau} \sum_{s=1}^{S} \sum_{\{s'|\tau_{s'}-\tau_s=j\}} \sum_{i=2}^{N} \xi_{s,s''}^k(i)}$$

$$s_v = \frac{\eta_j' + \sum_{k=1}^{K} \sum_{s=1}^{S} \sum_{\{s''\in H(s,v)\}} \sum_{i=2}^{N} \xi_{s,s''}^k(i)}{\sum_{j=0}^{1} \eta_j' + \sum_{k=1}^{K} \sum_{s=1}^{S} \sum_{\{s''\in H(s,1),H(s,0)\}} \sum_{i=2}^{N} \xi_{s,s''}^k(i)}$$

where $H(s,j) \equiv \{s'|s'\text{is exactly } j \text{ scale states away from } s\}$. Note that we do not normalize the Dirichlets, and omit the traditional minus one in the exponent: $\mathcal{D}(d_v^k|\{\eta_v^k\}) = \prod_{v=1}^{J_\tau}(d_v^k)^{\eta_v^k}$ and $\mathcal{D}(s_v|\{\eta_v'\}) = \prod_{v=0}^{1}(s_v)^{\eta_v'}$.

The M-Step updates $u^k$, $\sigma$, and $\vec{z}$ are coupled. Thus we arbitrarily pick an order to update them and as one is updated, its new values are used in the updates for the coupled parameter updates that follow it. In our experiments we updated in the following order: $\sigma$, $\vec{z}$, $u^k$. The other two parameters, $d_v^k$ and $s_v$, are completely decoupled.

## 4 Experiments with Laboratory and Speech Data

We have applied the CPM model to analyze several Liquid Chromatography - Mass Spectrometry (LC-MS) data sets from an experimental biology laboratory. Mass spectrometry technology is currently being developed to advance the field of proteomics [2, 3]. A mass spectrometer takes a sample as input, for example, human blood serum, and produces a measure of the abundance of molecules that have particular mass/charge ratios. In proteomics the molecules in question are small protein fragments. From the pattern of abundance values one can hope to infer which proteins are present and in what quantity. For protein mixtures that are very complex, such as blood serum, a sample preparation step is used to physically separate parts of the sample on the basis of some property of the molecules, for example, hydrophobicity. This separation spreads out the parts over time so that at each unique time point a less complex mixture is fed into the mass spectrometer. The result is a two-dimensional time series spectrum with mass/charge on one axis and time of input to the mass spectrometer on the other. In our experiments we collapsed the data at each time point to one dimension by summing together abundance values over all mass/charge values. This one-dimensional data is referred to as the Total Ion Count (TIC). We discuss alternatives to this in the last section. After alignment of the TICs, we assessed the alignment of the LC-MS data by looking at both the TIC alignments, and also the corresponding two-dimensional alignments of the non-collapsed data, which is where the true information lies.

The first data set was a set of 13 replicates, each using protein extracted from lysed *E. coli* cells. Proteins were digested and subjected to capillary-scale LC-MS coupled on-line to an ion trap mass spectrometer. First we trained the model with no smoothing (i.e., $\lambda = 0$) on the 13 replicates. This provided nice alignments when viewed in both the TIC space and the full two-dimensional space. Next we used leave-one-out cross-validation on six of the replicates in order to choose a suitable value for $\lambda$. Because the $u^k$ and $d_v^k$ are time series specific, we ran a restricted EM on the hold-out case to learn these parameters, holding the other parameters fixed at the values found from learning on the training set. Sixteen values of $\lambda$ over five orders of magnitude, and also zero, were used. Note that we did not include the regularization likelihood term in the calculations of hold-out likelihood. One of the non-zero values was found to be optimal (statistically significant at a p=0.05 level using a paired sample t-test to compare it to no smoothing). Visually, there did not appear to be a difference between no regularization and the optimal value of $\lambda$, in either the TIC space

or the full two-dimensional space. Figure 2 shows the alignments applied to the TICs and also the two-dimensional data, using the optimal value of $\lambda$.

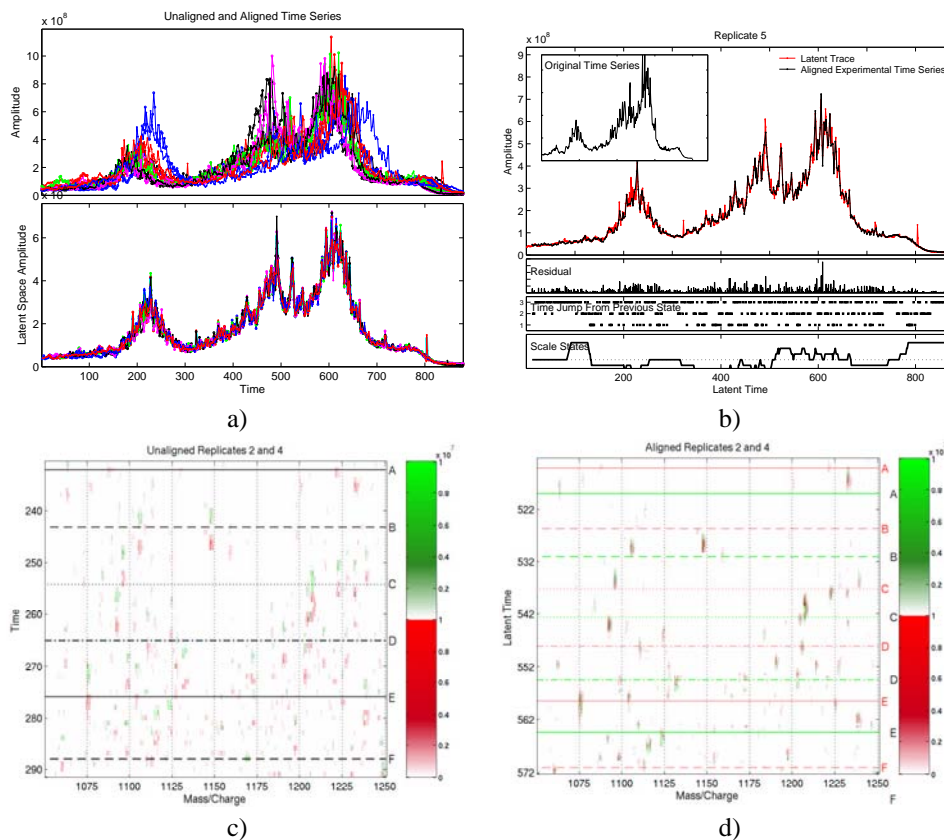

Figure 2: a) Top: 13 Replicate pre-processed TICs as described in Section 4), Bottom: same as top, but aligned with CPM (the learned latent trace is also shown). b) The fifth TIC replicate aligned to the learned latent trace (inset shows the original, unaligned). Below are three strips showing, from top-to-bottom, i) the error residual, ii) the number of time states moved between every two states in the Viterbi alignment, and iii) the local scaling applied at each point in the alignment. c) A portion of the two-dimensional LC-MS data from replicates two (in red) and four (in green). d) Same as c), but after alignment (the same one dimensional alignment was applied to every Mass/Charge value). Marker lines labeled A to F show how time in c) was mapped to latent time using the Viterbi alignment.

We also trained our model on five different sets of LC-MS data, each consisting of human blood serum. We used no smoothing and found the results visually similar in quality to the first data set.

To ensure convergence to a good local optimum and to speed up training, we pre-processed the LC-MS data set by coarsely aligning and scaling each time series as follows: We 1) translated each time series so that the center of mass of each time series was aligned to the median center of mass over all time series, 2) scaled the abundance values such that the sum of abundance values in each time series was equal to the median sum of abundance values over all time series.

We also used our model to align 10 speech signals, each an utterance of the same sentence

spoken by a different speaker. The short-time energy (using a 30ms Hanning window) was computed every 8ms for each utterance and the resulting vectors were used as the input to CPM for alignment. The smoothing parameter $\lambda$ was set to zero. For comparison, we also performed a linear warping of time with an offset. (i.e. each signal was translated so as to start at the same time, and the length of each signal was stretched or compressed so as to each occupy the same amount of time). Figure 1 shows the successful alignment of the speech signals by CPM and also the (unsuccessful) linear warp. Audio for this example can be heard at `http://www.cs.toronto.edu/~jenn/alignmentStudy`, which also contains some supplemental figures for the paper.

Initialization for EM training was performed as follows: $\sigma$ was set to 15% of the difference between the maximum and minimum values of the first time series. The latent trace was initialized to be the first observed time series, with Gaussian, zero-mean noise added, with standard deviation equal to $\sigma$. This was then upsampled by a factor of two by repeating every value twice in a row. The additional slack at either end of the latent trace was set to be the minimum value seen in the given time series. The $u^k$ were each set to one and the multinomial scale and state transition probabilities were set to be uniform.

## 5 Related Algorithms and Models

Our proposed CPM has many similarities to Input/Output HMMs (IOHMMs), also called Conditional HMMs [4]. IOHMMs extend standard HMMs [1] by conditioning the emission and transition probabilities on an observed input sequence. Each component of the output sequence corresponds to a particular component of the input. Training of an IOHMM is supervised — a mapping from an observed input sequence to output target sequence is learned. Our CPM also requires input and thus is also a type of conditional HMM. However, the input is unobserved (but crucially it is shared between all replicates) and hence learning is unsupervised in the CPM model. One could also take the alternative view that the CPM is simply an HMM with an extra set of parameters, the latent trace, that affect the emission probabilities and which are learned by the model.

The CPM is similar in spirit to Profile HMMs which have been used with great success for discrete, multiple sequence alignment, modeling of protein families and their conserved structures, gene finding [5], among others. Profile HMM are HMMs augmented by constrained-transition 'Delete' and 'Insert' states, with the former emitting no observations. Multiple sequences are provided to the Profile HMM during training and a summary of their shared statistical properties is contained in the resulting model. The development of Profile HMMs has provided a robust, statistical framework for reasoning about sets of related discrete sequence data. We put forth the CPM as a continuous data, conditional analogue.

Many algorithms currently used for aligning continuous time series data are variations of Dynamic Time Warping (DTW) [6], a dynamic programming based approach which originated in the speech recognition community as a robust distance measure between two time series. DTW works on pairs of time series, aligning one time series to a specified reference time series. DTW does not take in to account systematic variations in the amplitude of the signal. Our CPM can be viewed as a rich and robust extension of DTW that can be applied to many time series in parallel and which automatically uncovers the underlying template of the data.

## 6 Discussion and Conclusion

We have introduced a generative model for sets of continuous, time series data. By training this model one can leverage information contained in noisy, replicated experimental data,

and obtain a single, superior resolution 'fusion' of the data. We demonstrated successful use of this model on real data, but note that it could be applied to a wide range of problems involving time signals, for example, alignment of gene expression time profiles, alignment of temporal physiological signals, alignment of motion capture data, to name but a few.

Certain assumptions of the model presented here may be violated under different experimental conditions. For example, the Gaussian emission probabilities treat errors in large amplitudes in the same absolute terms as in smaller amplitudes, whereas in reality, it may be that the error scales with signal amplitude. Similarly, the penalty term $-\lambda \sum_{j=1}^{\tau-1}(z_{j+1} - z_j)^2$ does not scale with the amplitude; this might result in the model arbitrarily preferring a lower amplitude latent trace. (However, in practice, we did not find this to be a problem.)

One immediate and straight-forward extension to the model would be to allow the data at each time point to be a multi-dimensional feature vector rather than a scalar value. This could easily be realized by allowing the emission probabilities to be multi-dimensional. In this way a richer set of information could be used: either the raw, multi-dimensional feature vector, or some transformation of the feature vectors, for example, Principal Components Analysis. The rest of the model would be unchanged and each feature vector would move as a coherent piece. However, it might also be useful to allow different dimensions of the feature vector to be aligned differently. For example, with the LC-MS data, this might mean allowing different mass/charge peptides to be aligned differently at each time point. However, in its full generality, such a task would be extremely computational intense.

A perhaps more interesting extension is to allow the model to work with non-replicate data. For example, suppose one had a set of LC-MS experiments from a set of cancer patients, and also a set from normal persons. It would be desirable to align the whole set of time series and also to have the model tease out the differences between them. One approach is to consider the model to be semi-supervised - the model is told the class membership of each training example. Then each class is assigned its own latent trace, and a penalty is introduced for any disagreements between the latent traces. Care needs to be taken to ensure that the penalty plateaus after a certain amount of disagreement between latent trace points, so that parts of the latent trace which are truly different are able to whole-heartedly disagree. Assuming that the time resolution in the observed time series is sufficiently high, one might also want to encourage the amount of disagreement over time to be Markovian. That is, if the previous time point disagreed with the other latent traces, then the current point should be more likely to disagree.

# References

[1] Alan B. Poritz. Hidden markov models: A guided tour. In *Proceedings of the IEEE Conference on Acoustics, Speech and Signal Processing (ICASSP)*, pages 7–13. Morgan Kaufmann, 1988.

[2] Ruedi Aebersold and Matthias Mann. Mass spectrometry-based proteomics. *Nature*, 422:198–207, 2003.

[3] P. Kearney and P. Thibault. Bioinformatics meets proteomics - bridging the gap between mass spectrometry data analysis and cell biology. *Journal of Bioinformatics and Computational Biology*, 1:183–200, 2003.

[4] Yoshua Bengio and Paolo Frasconi. An input output HMM architecture. In G. Tesauro, D. Touretzky, and T. Leen, editors, *Advances in Neural Information Processing Systems*, volume 7, pages 427–434. The MIT Press, 1995.

[5] Richard Durbin, Sean R. Eddy, Anders Krogh, and Graeme Mitchison. *Biological Sequence Analysis: Probabilistic Models of Proteins and Nucleic Acids*. Cambridge Univ. Press, 2000. Durbin.

[6] H. Sakoe and S.Chiba. Dynamic programming algorithm for spoken word recognition. *Readings in Speech Recognition*, pages 159–165, 1990.
